# Penalized Principal Component Regression on Graphs for Analysis of Subnetworks

**Ali Shojaie**
Department of Statistics
University of Michigan
Ann Arbor, MI 48109
shojaie@umich.edu

**George Michailidis**
Department of Statistics and EECS
University of Michigan
Ann Arbor, MI 48109
gmichail@umich.edu

## Abstract

Network models are widely used to capture interactions among component of complex systems, such as social and biological. To understand their behavior, it is often necessary to analyze functionally related components of the system, corresponding to subsystems. Therefore, the analysis of subnetworks may provide additional insight into the behavior of the system, not evident from individual components. We propose a novel approach for incorporating available network information into the analysis of arbitrary subnetworks. The proposed method offers an efficient dimension reduction strategy using Laplacian eigenmaps with Neumann boundary conditions, and provides a flexible inference framework for analysis of subnetworks, based on a group-penalized principal component regression model on graphs. Asymptotic properties of the proposed inference method, as well as the choice of the tuning parameter for control of the false positive rate are discussed in high dimensional settings. The performance of the proposed methodology is illustrated using simulated and real data examples from biology.

## 1 Introduction

Simultaneous analysis of groups of system components with similar functions, or subsystems, has recently received considerable attention. This problem is of particular interest in high dimensional biological applications, where changes in individual components may not reveal the underlying biological phenomenon, whereas the combined effect of functionally related components could improve the efficiency and interpretability of results. This idea has motivated the method of *gene set enrichment analysis* (GSEA), along with a number of related methods [1, 2]. The main premise of this method is that by assessing the significance of sets rather than individual components (i.e. genes), interactions among them can be preserved, and more efficient inference methods can be developed. A different class of models (see e.g. [3, 4] and references therein) has focused on directly incorporating the network information in order to achieve better efficiency in assessing the significance of individual components.

These ideas have been combined in [5, 6], by introducing a model for incorporating the regulatory gene network, and developing an inference framework for analysis of subnetworks defined by biological pathways. In this frameworks, called NetGSA, a global model is introduced with parameters

for individual genes/proteins, and the parameters are then combined appropriately in order to assess the significance of biological pathways. However, the main challenge in applying NetGSA in real-world biological applications is the extensive computational time. In addition, the total number of parameters allowed in the model are limited by the available sample size $n$ (see Section 5).

In this paper, we propose a dimension reduction technique for networks, based on Laplacian eigenmaps, with the goal of providing an optimal low-dimensional projection for the space of random variables in each subnetwork. We then propose a general inference framework for analysis of subnetworks by reformulating the inference problem as a penalized principal regression problem on the graph. In Section 2, we review the Laplacian eigenmaps and establish their connection to principal component analysis (PCA) for random variables on a graph. Inference for significance of subnetworks is discussed in Section 3, where we introduce Laplacian eigenmaps with Neumann boundary conditions and present the group-penalized principal component regression framework for analysis of arbitrary subnetworks. Results of applying the new methodology to simulated and real data examples are presented in Section 4, and the results are summarized in Section 5.

## 2    Laplacian Eigenmaps

Consider $p$ random variables $X_i, i = 1, \ldots, p$ (e.g. expression values of genes) defined on nodes of an undirected (weighted) graph $\mathscr{G} = (V, E)$. Here $V$ is the set of nodes of $\mathscr{G}$ and $E \subseteq V \times V$ its edge set. Throughout this paper, we represent the edge set and the strength of associations among nodes through the adjacency matrix of the graph $A$. Specifically, $A_{ij} \geq 0$ and $i$ and $j$ are adjacent if the $A_{ij}$ (and hence $A_{ji}$) is non-zero. In this case we write $i \sim j$. Finally, we denote the observed values of the random variables by the $n \times p$ data matrix $X$.

The subnetworks of interest are defined based on additional knowledge about their attributes and functions. In biological applications, these subnetworks are defined by common biological function, co-regulation or chromosomal location. The objective of the current paper is to develop dimension reduction methods on networks, in order to assess the significance of *a priori* defined subnetworks (e.g. biological pathways) with minimal information loss.

### 2.1    Graph Laplacian and Eigenmaps

Laplacian eigenmaps are defined using the eigenfunctions of the graph Laplacian, which is commonly used in spectral graph theory, computer science and image processing. Applications based on Laplacian eigenmaps include image segmentation and the normalized cut algorithm of [7], spectral clustering [8, 9] and collaborative filtering [10].

The Laplacian matrix and its eigenvectors have also been used in biological applications. For example, in [11], the Laplacian matrix has been used to define a network-penalty for variable selection on graphs, and the interpretation of Laplacian eigenmaps as a Fourier basis was exploited in [12] to propose supervised and unsupervised classification methods.

Different definitions and representations have been proposed for the spectrum of a graph, and the results may vary depending on the definition of the Laplacian matrix (see [13] for a review). Here, we follow the notation in [13], and consider the *normalized* Laplacian matrix of the graph. To that end, let $D$ denote the diagonal degree matrix for $A$, i.e. $D_{ii} = \sum_j A_{ij} \equiv d_i$, and define the Laplacian matrix of the graph by $\mathscr{L} = D^{-1/2}(D - A)D^{-1/2}$, or alternatively

$$\mathscr{L}_{ij} = \begin{cases} 1 - \frac{A_{jj}}{d_j} & j = i, d_j \neq 0 \\ -\frac{A_{ij}}{\sqrt{d_i d_j}} & j \sim i \\ 0 & o.w. \end{cases}$$

It can be shown that [13] $\mathscr{L}$ is positive semidefinite with eigenvalues $0 = \lambda_0 \le \lambda_1 \le \ldots \le \lambda_{p-1} \le 2$. Its eigenfunctions are known as the spectrum of $\mathscr{G}$, and optimize the Rayleigh quotient

$$\frac{\langle g, \mathscr{L}g \rangle}{\langle g, g \rangle} = \frac{\sum_{i \sim j}(f(i)-f(j))^2}{\sum_j f(j)^2 d_j}, \tag{1}$$

It can be seen from (1), that the 0-eigenvalue of $\mathscr{L}$ is $g = D^{1/2}\mathbf{1}$, corresponding to the average over the graph $\mathscr{G}$. The first non-zero eigenvalue $\lambda_1$ is the harmonic eigenfunction of $\mathscr{L}$, which corresponds to the Laplace-Beltrami operator on Reimannian manifolds, and is given by

$$\lambda_1 = \inf_{f \perp D\mathbf{1}} \frac{\sum_{j \sim i}(f(i)-f(j))^2}{\sum_j f(j)^2 d_j}$$

More generally, denoting by $C_{k-1}$ the projection to the subspace of the first $k-1$ eigenfunctions,

$$\lambda_k = \inf_{f \perp DC_{k-1}} \frac{\sum_{j \sim i}(f(i)-f(j))^2}{\sum_j f(j)^2 d_j}.$$

## 2.2 Principal Component Analysis on Graphs

Previous applications of the graph Laplacian and its spectrum often focus on the properties of the graph; however, the connection to the probability distribution of the random variables on nodes of the graph has not been strongly emphasized. In graphical models, the undirected graph $\mathscr{G}$ among random variables corresponds naturally to a Markov random field [14]. The following result establishes the relationship between the Laplacian eigenmaps and the principal components of the random variables defined on the nodes of the graph, in case of Gaussian observations.

**Lemma 1.** *Let $X = (X_1, \ldots, X_p)$ be random variables defined on the nodes of graph $\mathscr{G} = (V, E)$ and denote by $\mathscr{L}$ and $\mathscr{L}^+$ the Laplacian matrix of $\mathscr{G}$ and its Moore-Penrose generalized inverse. If $X \sim N(0, \Sigma)$, then $\mathscr{L}$ and $\mathscr{L}^+$ correspond to $\Omega$ and $\Sigma$, respectively ($\Omega \equiv \Sigma^{-1}$). In addition, let $v_0, \ldots, v_{p-1}$ denote the eigenfunctions corresponding to eigenvalues of $\mathscr{L}$. Then $v_0, \ldots, v_{p-1}$ are the principal components of $X$, with $v_0$ corresponding to the leading principal component.*

*Proof.* For Gaussian random variables, the inverse covariance (or precision) matrix has the same non-zero pattern as the adjacency matrix of the graph, i.e. for $i \ne j$, $\Omega_{ij} = 0$ iff $A_{ij} = 0$. Moreover, $\Omega_{ii} = \tau_i^{-2}$, where $\tau_i^2$ is the partial variance of $X_i$ (see e.g. [15]). However, using the conditional autoregression (CAR) representation of Gaussian Markov random fields [16], we can write

$$\mathbb{E}(X_i | X_{-i}) = \sum_{j \sim i} c_{ij} X_j \tag{2}$$

where $-i \equiv \{1 \ldots p\} \backslash i$ and $C = [c_{ij}]$ has the same non-zero pattern as the adjacency matrix of the graph $A$, and amounts to a proper probability distribution for $X$. In particular, by Brook's Lemma [16] it follows from (2) that $f_X(x) \propto \exp\{-1/2x^{\mathsf{T}}(0, T^{-1}(I_p - C))x\}$, where $T = \text{diag}[\tau_i^2]$. Therefore, $\Omega = T^{-1}(I_p - C)$ and hence $(I_p - C)$ should be PD.

However, since $\mathscr{L} = I_p - D^{-1/2}AD^{-1/2}$ is PSD, we can set $C = D^{-1/2}AD^{-1/2} - \zeta I$ for any $\zeta > 0$. In other words, $(I_p - C) = \mathscr{L} + \zeta I_p$, which implies that $\tilde{\mathscr{L}} \equiv \mathscr{L} + \zeta I_p = T\Omega$, and hence $\tilde{\mathscr{L}}^{-1} = \Sigma T^{-1}$. Taking limit as $\zeta \to 0$, it follows that $\mathscr{L}$ and $\mathscr{L}^+$ correspond to $\Omega$ and $\Sigma$, respectively.

The second part follows directly from the above connection between $\tilde{\mathscr{L}}^{-1}$ and $\Sigma$. In particular, suppose, without loss of generality, that $\tau_i^2 = 1$. Then, it is easily seen that the principal components of $X$ are given by eigenfunctions of $\tilde{\mathscr{L}}^{-1}$, which are in turn equal to the eigenfunctions of $\tilde{\mathscr{L}}$ with the ordering of the eigenvalues reversed. However, since eigenfunctions of $\mathscr{L} + \zeta I_p$ and $\mathscr{L}$ are equal, the principal components of $X$ are obtained from eigenfunctions of $\mathscr{L}$. $\square$

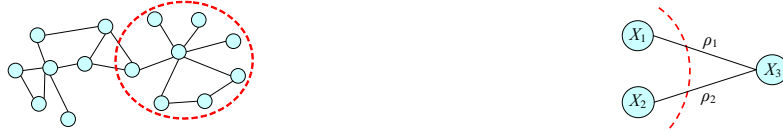

Figure 1: Left: A simple subnetwork of interest, marked with the dotted circle. Right: Illustration of the Neumann random walk, the dotted curve indicates the boundary of the subnetwork.

*Remark* 2. An alternative justification for the above result, for general probability distributions defined on graphs, can be given by assuming that the graph represents "similarities" among random variables and using an optimal embedding of graph $\mathcal{G}$ in a lower dimensional Euclidean space[1]. In the case of one dimensional embedding, the goal is to find an embedding $v = (v_1, \ldots, v_p)^\mathsf{T}$ that preserves the distances among the nodes of the graph. The objective function of the embedding problem is then given by $Q = \sum_{i,j} (v_i - v_j)^2 A_{ij}$, or alternatively $Q = 2v^\mathsf{T}(D - A)v$ [17]. Thus, the optimal embedding is found by solving $\mathrm{argmin}_{v^\mathsf{T} Dv = 1} v^\mathsf{T}(D - A)v$. Setting $u = D^{1/2}v$, this is solved by finding the eigenvector corresponding to the smallest eigenvalue of $\mathscr{L}$.

Lemma 1 provides an efficient dimension reduction framework that summarizes the information in the entire network into few feature vectors. Although the resulting dimension reduction method can be used efficiently in classification (as in [12]), the eigenfunctions of $\mathcal{G}$ do not provide any information about significance of arbitrary subnetworks, and therefore cannot be used to analyze the changes in subnetworks. In the next section, we introduce a restricted version of Laplacian eigenmaps, and discuss the problem of analysis of subnetworks.

## 3   Analysis of Subnetworks and PCR on Graph (GPCR)

In [5], the authors argue that to analyze the effect of subnetworks, the test statistic needs to represent the pure effect of the subnetwork, without being influenced by external nodes, and propose an inference procedure based on mixed linear models to achieve this goal. However, in order to achieve dimension reduction, we need a method that only incorporates *local* information at the level of each subnetwork, and possibly its neighbors (see the left panel of Figure 1).

Using the connection of the Laplace operator in Reimannian manifolds to heat flow (see e.g. [17]), the problem of analysis of arbitrary subnetworks can be reformulated as a heat equation with boundary conditions. It then follows that in order to assess the "effect" of each subnetwork, the appropriate boundary conditions should block the flow of heat at the boundary of the set. This corresponds to insulating the boundary, also known as the *Neumann boundary condition*. For the general heat equation $\tau(v, x)$, this boundary condition is given by $\frac{\partial \tau}{\partial v}(x) = 0$ at each boundary point $x$, where $v$ is the normal direction orthogonal to the tangent hyperplane at $x$.

The eigenvalues of subgraphs with boundary conditions are studied in [13]. In particular, let $S$ be any (connected) subnetwork of $\mathcal{G}$, and denote by $\delta S$ the boundary of $S$ in $\mathcal{G}$. The Neumann boundary condition states that for every $x \in \delta S$, $\sum_{y:\{x,y\}\in\delta S} (f(x) - f(y)) = 0$.

The Neumann eigenfunctions of $S$ are then the optimizers of the restricted Rayleigh quotient

$$\lambda_{S,i} = \inf_f \sup_{g \in C_{i-1}} \frac{\sum_{\{t,u\}\in S \cup \delta S} (f(t) - f(u))^2}{\sum_{t \in S} (f(t) - g(t))^2 d_t}$$

where $C_{i-1}$ is the projection to the space of previous eigenfunctions.

In [13], a connection between the Neumann boundary conditions and a reflected random walk on the graph is established, and it is shown that the Neumann eigenvectors can be alternatively calculated from the eigenvectors of the transition probability matrix of this reflected random walk, also known as the Neumann random walk (see [13] for additional details). Here, we generalize this idea to weighted adjacency matrices.

Let $\tilde{P}$ and $P$ denote the transition probability matrix of the reflected random walk, and the original random walk defined on $\mathcal{G}$, respectively. Noting that $P = D^{-1}A$, we can extend the results in [13] as follows. For the general case of weighted graphs, define the transition probability matrix of the reflected random walk by

$$\tilde{P}_{ij} = \begin{cases} P_{ij} & j \sim i, i, j \in S \\ P_{ij} + \frac{A_{ik}A_{kj}}{d_i d_k'} & j \sim k \sim i, k \notin S \\ 0 & o.w. \end{cases} \tag{3}$$

where $d_k' = \sum_{i \sim k, i \in S} A_{ki}$ denotes the degree of the node $k$ in $S$. Then, the Neumann eigenvalues are given by $\lambda_i = 1 - \kappa_i$, where $\kappa_i$ is the $i$th eigenvalue of $\tilde{P}$.

*Remark* 3. The connection with the Neumann random walk also sheds light into the effect of the proposed boundary condition on the joint probability distribution of the random variables on the graph. To illustrate this, consider the simple graph in the right panel of Figure 1. For the moment, suppose that the random variables $X_1, X_2, X_3$ are Gaussian, and the edges from $X_1$ and $X_2$ to $X_3$ are directed. As discussed in [5], the joint probability distribution of the random variables on the graph is then given by linear structural equation models:

$$\begin{array}{rcl} X_1 & = & \gamma_1 \\ X_2 & = & \gamma_2 \\ X_3 & = & \rho_1 X_1 + \rho_1 X_2 \end{array} \qquad \Rightarrow \qquad Y = \Lambda \gamma, \qquad \Lambda = \begin{pmatrix} 1 & 0 & 0 \\ 0 & 1 & 0 \\ \rho_1 & \rho_2 & 1 \end{pmatrix}$$

Then, the conditional probability distribution of $X_1$ and $X_2$ given $X_3$, is Gaussian, with the inverse covariance matrix given by

$$\begin{pmatrix} 1 + \rho_1^2 & \rho_1 \rho_2 \\ \rho_1 \rho_2 & 1 + \rho_2^2 \end{pmatrix} \tag{4}$$

A comparison between (3) and (4) then reveals that the proposed Neumann random walk corresponds to conditioning on the boundary variables, if the edges going from the set $S$ to its boundary are directed. The reflected random walk, for the original problem, therefore corresponds to first setting all the influences from other nodes in the graph to nodes in the set $S$ to zero (resulting in directed edges) and then conditioning on the boundary variables. Therefore, the proposed method offers a compromise compared to the full model of [5], based on local information at the level of each subnetwork.

### 3.1 Group-Penalized PCR on Graph

Using the Neumann eigenvectors of subnetworks, we now define a principal component regression on graphs, which can be used to analyze the significance of subnetworks. Let $\mathcal{N}_j$ denote the $|S_j| \times m_j$ matrix of the $m_j$ smallest Neumann eigenfunctions for subgraph $S_j$. Also, let $X^{(j)}$ be the $n \times |S_j|$ matrix of observations for the $j$-th subnetwork. An $m_j$-dimensional projection of the original data matrix $X^{(j)}$ is then given by $\tilde{X}^{(j)} = X^{(j)}N_j$. Different methods can be used in order to determine the number of eigenfunctions $m_j$ for each subnetwork. A simple procedure determines a predefined threshold for the proportion of variance explained by each eigenfunction. These proportions can be determined by considering the reciprocal of Neumann eigenvalues (ignoring the 0-eigenvalue). To simplify the presentation, here we assume $m_j = m, \forall j$.

The significance of subnetwork $S_j$ is a function of the combined effect of all the nodes, captured by the transformed data matrix $\tilde{X}^{(j)}$. This can be evaluated by forming a multivariate ANOVA (MANOVA) model. Formally, let $y$ be the $mn \times 1$ vector of observations obtained by stacking all the transformed data matrices $\tilde{X}^{(j)}$. Also, let $\mathscr{X}$ be the $mn \times Jmr$ design matrix corresponding to the experimental settings, where $r$ is the number of parameters used to model experimental conditions, and $\beta$ be the vector of regression coefficients. For simplicity, here we focus on the case of a two-class inference problem (e.g. treatment vs. control). Extensions to more general experimental settings follow naturally and are discussed in Section 5.

To evaluate the combined effect of each subnetwork, we impose a group penalty on the coefficient of the regression of $y$ on the design matrix $\mathscr{X}$. In particular, using the group lasso penalty [18], we estimate the significance of the subnetwork by solving the following optimization problem[2]

$$\underset{\beta}{\text{argmin}} \left\{ n^{-1} \| y - \sum_{j=1}^{J} \mathscr{X}^{(j)} \beta^{(j)} \|_2^2 + \gamma \sum_{j=1}^{J} \| \beta^{(j)} \|_2 \right\} \tag{5}$$

where $J$ is the total number of subnetworks considered and $\mathscr{X}^{(j)}$ and $\beta^{(j)}$ denote the columns of $\mathscr{X}$, and entries of $\beta$ corresponding to the subnetwork $j$, respectively.

In equation (5), $\gamma$ is the tuning parameter and is usually determined by performing k-fold cross validation or evaluation on independent data sets. However, since the goal of our analysis is to determine the significance of subnetworks, $\gamma$ should be determined so that the *probability of false positives* is controlled at a given significance level $\alpha$. Here we adapt the approach in [20] and determine the optimal value of $\gamma$ so that the family-wise error rate (FWER) in repeated sampling with replacement (bootstrap) is controlled at the level $\alpha$. Specifically, let $q_\gamma^i$ be the total number of subnetworks considered significant based on the value of $\gamma$ in the $i$th bootstrap sample. Let $\pi$ be the threshold for selection of variables as significant. In other words, if $P_i^{(j)}$ is the probability of selecting the coefficients corresponding to subnetwork $j$ in the $i$th bootstrap sample, the subnetwork $j$ is considered significant if $\max_\gamma P_i^{(j)} \geq \pi$. Using this method, we select $\gamma$ such that $q_\gamma^i = \sqrt{(2\pi - 1)\alpha p}$.[3]

The following result shows that the proposed methodology correctly selects the significant subnetworks, while controlling FWER at level $\alpha$. We begin by introducing some additional notations and assumptions. We assume the columns of design matrix $\mathscr{X}$ are normalized so that $n^{-1} \mathscr{X}_i^\top \mathscr{X}_i = 1$, Throughout this paper, we consider the case where the total number of nodes in the graph $p$, and the number of design parameters $r$ are allowed to diverge (the $p \gg n$ setting). In addition, let $s$ be the total number of non-zero elements in the true regression vector $\beta$.

**Theorem 4.** *Suppose that $m, n \geq 1$ and there exists $\eta \geq 1$ and $t \geq s \geq 1$ such that $n^{-1} \mathscr{X}^\top \mathscr{X}_{ij} \leq (7\eta t)^{-1}$ for all $i \neq j$. Also suppose that for $j \neq k$, the transformed random variables $\tilde{X}^{(j)}$ and $\tilde{X}^{(k)}$ are independent. If the tuning parameter $\gamma$ is selected such that such that $q_\gamma = \sqrt{(2\pi - 1)\alpha r p}$,*

*(i) there exists $\zeta = \zeta(n, p) > 0$ such that $\zeta \to 0$ as $n \to \infty$ and with probability at least $1 - \zeta$ the significant subnetworks are correctly selected with high probability,*

*(ii) the family-wise error rate is controlled at the level $\alpha$.*

*Outline of the Proof.* First note that the MANOVA model presented above can be reformulated as a multi-task learning problem [21]. Upon establishing the fact that for the proposed tuning parameter $\gamma \sim \sqrt{\log p / (nm^{3/2})}$, it follows from the results in [22] that for each bootstrap sample, there exists $\varepsilon = \varepsilon(n) > 0$ such that with probability at least $1 - (rp)^{-\varepsilon}$ the significant subnetworks are correctly selected. Thus if $\pi \leq 1 - (rp)^{-\varepsilon}$, the coefficients for significant subnetworks are included in the final

model with hight probability. In particular, it can be shown that $\zeta = \Phi\{\sqrt{B}(1-(rp)^{-\varepsilon}-\pi)/2\}$, where $B$ is the number of bootstrap samples and $\Phi$ is the cumulative normal distribution. This proves the first claim.

Next, note that the normality assumption, and the fact that the eigenfunctions within each sub-network are orthogonal, imply that for each $j$, $\tilde{X}_i^{(j)}, i = 1, \ldots, m$ are independent. Moreover, the assumption of independence of $\tilde{X}^{(j)}$ and $\tilde{X}^{(k)}$ for $j \neq k$ implies that the values of $y$ are independent realizations of i.i.d standard normal random variables. On the other hand, the KarushKuhnTucker conditions for the optimization problem in (5) imply that $\beta^{(j)} \neq 0$ iff $(nm)^{(-1)}\langle (y - \mathscr{X}\beta), \mathscr{X}^{(j)}\rangle = \text{sgn}\,(\hat{\beta}^{(j)})\gamma$, where $\langle x, y\rangle$ denotes their inner product. It is hence clear that $1_{[\beta^{(j)} \neq 0]}$ are exchangeable. Combining this with the first part of the theorem, the claim follows from Theorem 1 of [20]. $\square$

*Remark* 5. The main assumption of Theorem 4 is the independence of the variables in different sub-networks. Although this is not satisfied in general problems, it may be satisfied by the conditioning argument of Remark 3. It is possible to further relax this assumption using an argument similar to Theorem 2 of [20], but we do not pursue this here.

## 4  Experiments

We illustrate the performance of the proposed method using simulated data motivated by biological applications, as well as a real data application based on gene expression analysis. In the simulation, we generate a small network of 80 nodes (genes), with 8 subnetworks. The random variables (expression levels of genes) are generated according to a normal distribution with mean $\mu$. Under the null hypothesis, $\mu_{null} = 1$ and the association weight $\rho$ for all edges of the network is set to 0.2. The setting of parameters under the alternative hypothesis are given in Table 1, where $\mu_{alt} = 3$. These settings are illustrated in the left panel of Figure 2. Table 1 also includes the estimated powers of the tests for subnetworks based on 200 simulations with $n = 50$ observations. It can be seen that the proposed GPCR method offers improvements over GSEA [1], especially in case of subnetworks 3 and 6. However, it results in a less accurate inference compared to NetGSA [5].

In [5], the pathways involved in Galactose utilization in yeast were analyzed based on the data from [23], and the performances of the NetGSA and GSEA methods were compared. The interactions among genes, along with significance of individual genes (based on single gene analysis) are given in the right panel of Figure 2, and the results of significance analysis based on NetGSA, GSEA and the proposed GPCR are given in Table 2. As in the simulated example, the results of this analysis indicate that GPCR results in improved efficiency over GSEA, while failing to detect the significance of some of the pathways detected by NetGSA.

## 5  Conclusion

We proposed a principal component regression method for graphs, called GPCR, using Laplacian eigenmaps with Neumann boundary conditions. The proposed method offers a systematic approach

Table 1: Parameter settings under the alternative and estimated powers for the simulation study.

| Subnet | Parameter Setting | | Estimated Powers | | | Subnet | Parameter Setting | | Estimated Powers | | |
|---|---|---|---|---|---|---|---|---|---|---|---|
| | % $\mu_{alt}$ | $\rho$ | NetGSA | GPCR | GSEA | | % $\mu_{alt}$ | $\rho$ | NetGSA | GPCR | GSEA |
| 1 | 0.05 | 0.2 | 0.02 | 0.08 | 0.01 | 5 | 0.05 | 0.6 | 0.94 | 0.41 | 0.12 |
| 2 | 0.20 | 0.2 | 0.03 | 0.21 | 0.02 | 6 | 0.20 | 0.6 | 1.00 | 0.61 | 0.15 |
| 3 | 0.50 | 0.2 | 1.00 | 0.65 | 0.27 | 7 | 0.50 | 0.6 | 1.00 | 0.99 | 0.97 |
| 4 | 0.80 | 0.2 | 1.00 | 0.81 | 0.90 | 8 | 0.80 | 0.6 | 1.00 | 0.99 | 1.00 |

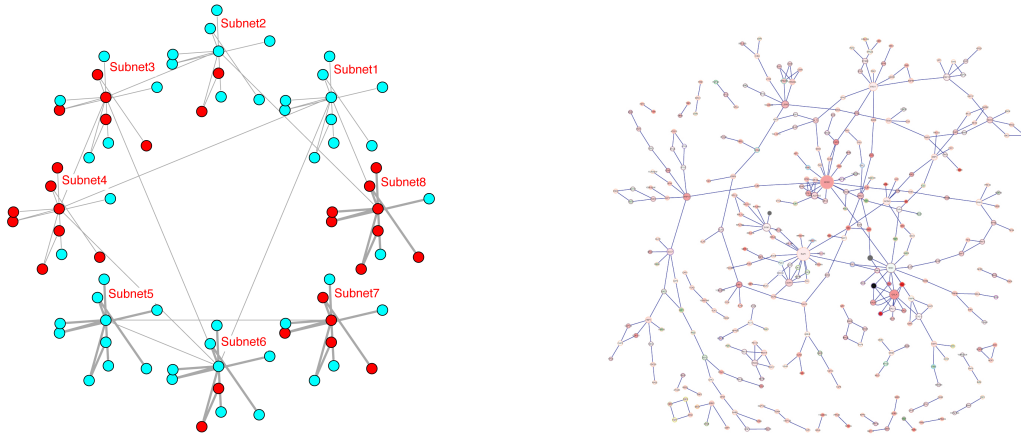

Figure 2: Left: Setting of the simulation parameters under the alternative hypothesis. Right: Network of yeast genes involved in Galactose utilization.

for dimension reduction in networks, with *a priori* defined subnetworks of interest. It can also incorporate both weighted and unweighted adjacency matrices and can be easily extended to analyzing complex experimental conditions through the framework of linear models. This method can also be used in longitudinal and time-course studies.

Our simulation studies, and the real data example indicate that the proposed GPCR method offers significant improvements over the methods of gene set enrichment analysis (GSEA). However, it does not achieve optimal powers in comparison to NetGSA. This difference in power may be attributable to the mechanism of incorporating the network information in the two methods: while NetGSA incorporates the full network information, GPCR only account for local network information, at the level of each subnetwork, and restricts the interactions with the rest of the network based on the Neumann boundary condition. However, the most computationally involved step in Net-GSA requires $O(p^3)$ operation, whereas the computational cost of GPCR is $O(m^3)$. It is clear that since $m \ll p$ in most applications, GPCR could result in significant improvement in terms of computational time and memory requirements for analysis of high dimensional networks. In addition, NetGSA requires that $r < n$, whilst the dimension reduction and the penalization of the proposed GPCR removes the need for any such restriction and facilitates the analysis of complex experiments in the settings with small sample sizes.

## Acknowledgments

Funding for this work was provided by NIH grants 1RC1CA145444-0110 and 5R01LM010138-02.

Table 2: Significance of pathways in Galactose utilization.

| PATHWAY | Size | NetGSA | GPCR | GSEA | PATHWAY | Size | NetGSA | GPCR | GSEA |
|---|---|---|---|---|---|---|---|---|---|
| rProtein Synthesis | 28 | ✓ | | | Sugar Transport | 2 | | | |
| Glycolytic Enzymes | 16 | | | | Glycogen Metabolism | 12 | | | |
| RNA Processing | 75 | | | | Stress | 12 | ✓ | ✓ | |
| Fatty Acid Oxidation | 7 | ✓ | ✓ | | Metal Uptake | 4 | | | |
| O2 Stress | 13 | | | | Respiration | 9 | ✓ | | |
| Mating, Cell Cycle | 58 | | | | Gluconeogenesis | 7 | | | |
| Vesicular Transport | 19 | | | | Galactose Utilization | 12 | ✓ | ✓ | ✓ |
| Amino Acid Synthesis | 30 | | | | | | | | |

## Footnotes

[1] For unweighted graphs, this justification was given by [17], using the unnormlized Laplacian matrix.

[2]The problem in (5) can be solved using the R-package `grplasso` [19].

[3]Additional details for this method are given in [20], but are excluded here due to space limitations.

# References

[1] A. Subramanian, P. Tamayo, V.K. Mootha, S. Mukherjee, B.L. Ebert, M.A. Gillette, A. Paulovich, S.L. Pomeroy, T.R. Golub, E.S. Lander, et al. Gene set enrichment analysis: A knowledge-based approach for interpreting genome-wide expression profiles. *Proceedings of the National Academy of Sciences*, 102(43):15545–15550, 2005.

[2] B. Efron and R. Tibshirani. On testing the significance of sets of genes. *Annals of Applied Statistics*, 1(1):107–129, 2007.

[3] T. Ideker, O. Ozier, B. Schwikowski, and A.F. Siegel. Discovering regulatory and signalling circuits in molecular interaction networks. *Bioinformatics*, 18(1):S233–S240, 2002.

[4] Zhi Wei and Li Hongzhe. A markov random field model for network-based analysis of genomic data. *Bioinformatics*, 2007.

[5] A. Shojaie and G. Michailidis. Analysis of gene sets based on the underlying regulatory network. *Journal of Computational Biology*, 16(3):407–426, 2009.

[6] A. Shojaie and G. Michailidis. Network enrichment analysis in complex experiments. *Statisitcal Applications in Genetics and Molecular Biology*, 9(1), Article 22, 2010.

[7] J. Shi and J. Malik. Normalized cuts and image segmentation. *IEEE Transactions on pattern analysis and machine intelligence*, 22(8):888–905, 2000.

[8] M. Saerens, F. Fouss, L. Yen, and P. Dupont. The principal components analysis of a graph, and its relationships to spectral clustering. *Machine Learning: ECML 2004*, pages 371–383, 2004.

[9] A.Y. Ng, M.I. Jordan, and Y. Weiss. On spectral clustering: Analysis and an algorithm. *Advances in neural information processing systems*, 2:849–856, 2002.

[10] F. Fouss, A. Pirotte, J.M. Renders, and M. Saerens. A novel way of computing dissimilarities between nodes of a graph, with application to collaborative filtering and subspace projection of the graph nodes. In *European Conference on Machine Learning Proceedings, ECML*, 2004.

[11] C. Li and H. Li. Variable Selection and Regression Analysis for Graph-Structured Covariates with an Application to Genomics. *Annals of Applied Statistics,* in press, 2010.

[12] F. Rapaport, A. Zinovyev, M. Dutreix, E. Barillot, and J.P. Vert. Classification of microarray data using gene networks. *BMC bioinformatics*, 8(1):35, 2007.

[13] F.R.K. Chung. *Spectral graph theory*. American Mathematical Society, 1997.

[14] S.L. Lauritzen. *Graphical models*. Oxford Univ Press, 1996.

[15] H. Rue and L. Held. *Gaussian Markov random fields: theory and applications*. Chapman & Hall, 2005.

[16] J. Besag. Spatial interaction and the statistical analysis of lattice systems. *Journal of the Royal Statistical Society. Series B (Methodological)*, 36(2):192–236, 1974.

[17] M. Belkin and P. Niyogi. Laplacian eigenmaps and spectral techniques for embedding and clustering. *Advances in neural information processing systems*, 1:585–592, 2002.

[18] M. Yuan and Y. Lin. Model selection and estimation in regression with grouped variables. *Journal of Royal Statistical Society. Series B Statistical Methodology*, 68(1):49, 2006.

[19] L. Meier, S. Van de Geer, and P. Buhlmann. The group lasso for logistic regression. *Journal of Royal Statistical Society. Series B Statistical Methodology*, 70(1):53, 2008.

[20] N. Meinshausen and P. Bühlmann. Stability selection. *Preprint, arXiv*, 809, 2009.

[21] A. Argyriou, T. Evgeniou, and M. Pontil. Convex multi-task feature learning. *Machine Learning*, 73(3):243–272, 2008.

[22] K. Lounici, M. Pontil, A.B. Tsybakov, and S. van de Geer. Taking Advantage of Sparsity in Multi-Task Learning. *Preprint, arXiv*, 903, 2009.

[23] T. Ideker, V. Thorsson, J.A. Ranish, R. Christmas, J. Buhler, J.K. Eng, R. Bumgarner, D.R. Goodlett, R. Aebersold, and L. Hood. Integrated genomic and proteomic analyses of a systematically perturbed metabolic network. *Science*, 292(5518):929, 2001.

